# Group Orthogonal Matching Pursuit for Variable Selection and Prediction

**Aurélie C. Lozano, Grzegorz Świrszcz, Naoki Abe**
IBM Watson Research Center,
1101 Kitchawan Road,
Yorktown Heights NY 10598,USA
{aclozano,swirszcz,nabe}@us.ibm.com

## Abstract

We consider the problem of variable group selection for least squares regression, namely, that of selecting groups of variables for best regression performance, leveraging and adhering to a natural grouping structure within the explanatory variables. We show that this problem can be efficiently addressed by using a certain greedy style algorithm. More precisely, we propose the Group Orthogonal Matching Pursuit algorithm (Group-OMP), which extends the standard OMP procedure (also referred to as "forward greedy feature selection algorithm" for least squares regression) to perform stage-wise group variable selection. We prove that under certain conditions Group-OMP can identify the correct (groups of) variables. We also provide an upperbound on the $l_\infty$ norm of the difference between the estimated regression coefficients and the true coefficients. Experimental results on simulated and real world datasets indicate that Group-OMP compares favorably to Group Lasso, OMP and Lasso, both in terms of variable selection and prediction accuracy.

## 1  Introduction

We address the problem of variable selection for regression, where a natural grouping structure exists within the explanatory variables, and the goal is to select the correct group of variables, rather than the individual variables. This problem arises in many situations (e.g. in multifactor ANOVA, generalized additive models, time series data analysis, where lagged variables belonging to the same time series may form a natural group, gene expression analysis from microarrays data, where genes belonging to the same functional cluster may be considered as a group). In these settings, selecting the right groups of variables is often more relevant to the subsequent use of estimated models, which may involve interpreting the models and making decisions based on them.

Recently, several methods have been proposed to address this variable group selection problem, in the context of linear regression [12, 15]. These methods are based on extending the Lasso formulation [8] by modifying the $l_1$ penalty to account for the group structure. Specifically, Yuan & Lin [12] proposed the Group Lasso, which solves $\arg\min_\beta \frac{1}{2} \left( \|y - \sum_{j=1}^J X_{G_j}\beta_{G_j}\|^2 + \lambda \sum_{j=1}^J \|\beta_{G_j}\|_2 \right)$, where $X_{G_1}, \ldots, X_{G_J}$ are the natural groupings within the variables of $X$ and $\beta_{G_j}$ are the coefficient vectors for variables in groups $G_j$. Zhao et al [15] considered a more general penalty class, the Composite Absolute Penalties family $T(\beta) = \sum_{j=1}^J \|\beta_j\|_{l_j}^{l_0}$, of which the Group Lasso penalty is a special instance. This development opens up a new direction of research, namely that of extending the existing regression methods with variable selection to the variable group selection problem and investigating to what extent they carry over to the new scenario.

The present paper establishes that indeed one recent advance in variable selection methods for regression, "forward greedy feature selection algorithm", also known as the Orthogonal Matching

Pursuit (OMP) algorithm in the signal processing community [5], can be generalized to the current setting of group variable selection. Specifically we propose the "Group Orthogonal Matching Pursuit" algorithm (*Group-OMP*), which extends the OMP algorithm to leverage variable groupings, and prove that, under certain conditions, Group-OMP can identify the correct (groups of) variables when the sample size tends to infinity. We also provide an upperbound on the $l_\infty$ norm of the difference between the estimated regression coefficients and the true coefficients. Hence our results generalize those of Zhang [13], which established consistency of the standard OMP algorithm. A key technical contribution of this paper is to provide a condition for Group-OMP to be consistent, which generalizes the "Exact Recovery Condition" of [9](Theorem 3.1) stated for OMP under the noiseless case. This result should also be of interest to the signal processing community in the context of block-sparse approximation of signals. We also conduct empirical evaluation to compare the performance of Group-OMP with existing methods, on simulated and real world datasets. Our results indicate that Group-OMP favorably compares to the Group Lasso, OMP and Lasso algorithms, both in terms of the accuracy of prediction and that of variable selection. Related work include [10, 3] using OMP for simultaneous sparse approximation, [11] showing that standard MP selects features from correct groups, and [4] that consider a more general setting than ours.

The rest of the paper is organized as follows. Section 2 describes the proposed Group-OMP procedure. The consistency results are then stated in Section 3. The empirical evaluation results are presented in Section 4. We conclude the paper with some discussions in Section 5.

## 2 Group Orthogonal Matching Pursuit

Consider the general regression problem $y = X\bar{\beta} + \nu$, where $y \in \mathbb{R}^n$ is the response vector, $X = [f_1, \ldots, f_d] \in \mathbb{R}^{n \times d}$ is the matrix of feature (or variable) vectors $f_j \in \mathbb{R}^n$, $\bar{\beta} \in \mathbb{R}^d$ is the coefficient vector and $\nu \in \mathbb{R}^n$ is the noise vector. We assume that the noise components $\nu_i$, $i = 1, \ldots, n$, are independent Gaussian variables with mean 0 and variance $\sigma^2$. For any $G \subset \{1, \ldots, d\}$ let $X_G$ denote the restriction of $X$ to the set of variables, $\{f_j, j \in G\}$, where the colums $f_j$ are arranged in ascending order. Similarly for any vector $\beta \in \mathbb{R}^d$ of regression coefficients, denote $\beta_G$ its restriction to $G$, with reordering in ascending order. Suppose that a natural grouping structure exists within the variables of $X$ consisting of $J$ groups $X_{G_1}, \ldots, X_{G_J}$, where $G_i \subset \{1, \ldots, d\}$, $G_i \cap G_j = \emptyset$ for $i \neq j$ and $X_{G_i} \in \mathbb{R}^{n \times d_j}$. Then, the above regression problem can be decomposed with respect to the groups, i.e. $y = \sum_{j=1}^J X_{G_j}\bar{\beta}_{G_j} + \nu$, where $\bar{\beta}_{G_j} \in \mathbb{R}^{d_j}$. Furthermore, to simplify the exposition, assume that each $X_{G_j}$ is orthonormalized, i.e. $X_{G_j}^* X_{G_j} = I_{d_j}$.

Given $\beta \in \mathbb{R}^d$ let $\text{supp}(\beta) = \{j : \beta_j \neq 0\}$. For any such $G$ and $v \in \mathbb{R}^n$, denote by $\hat{\beta}_X(G, v)$ the coefficients resulting from applying ordinary least squares (OLS) with non-zero coefficients restricted to $G$, i.e., $\hat{\beta}_X(G, v) = \arg\min_{\beta \in \mathbb{R}^d} \|X\beta - v\|_2^2$ subject to $\text{supp}(\beta) \subset G$. Given the above setup, the *Group-OMP* procedure we propose is described in Figure 1, which extends the OMP procedure to deal with group selection. Note that this procedure picks the best group in each iteration, with respect to reduction of the residual error, and it then re-estimates the coefficients, $\beta^{(k)}$, as in OMP. We recall that this re-estimation step is what distinguishes OMP, and our group version, from standard boosting-like procedures.

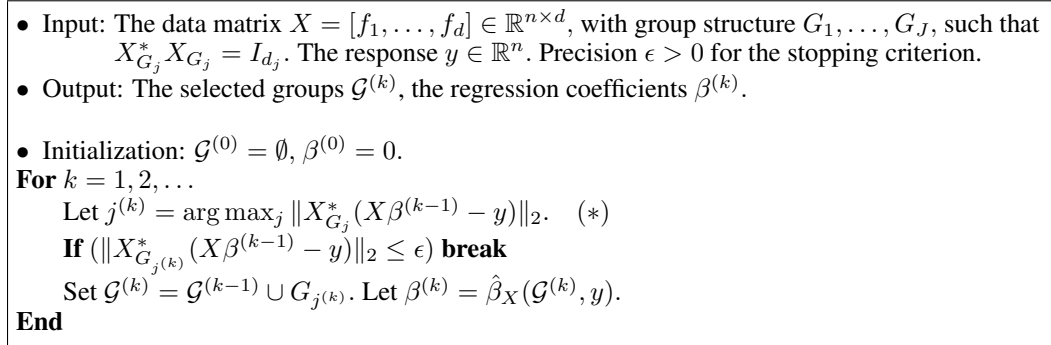

- Input: The data matrix $X = [f_1, \ldots, f_d] \in \mathbb{R}^{n \times d}$, with group structure $G_1, \ldots, G_J$, such that $X_{G_j}^* X_{G_j} = I_{d_j}$. The response $y \in \mathbb{R}^n$. Precision $\epsilon > 0$ for the stopping criterion.
- Output: The selected groups $\mathcal{G}^{(k)}$, the regression coefficients $\beta^{(k)}$.

- Initialization: $\mathcal{G}^{(0)} = \emptyset$, $\beta^{(0)} = 0$.

**For** $k = 1, 2, \ldots$
    Let $j^{(k)} = \arg\max_j \|X_{G_j}^*(X\beta^{(k-1)} - y)\|_2$.   (∗)
    **If** $(\|X_{G_{j^{(k)}}}^*(X\beta^{(k-1)} - y)\|_2 \leq \epsilon)$ **break**
    Set $\mathcal{G}^{(k)} = \mathcal{G}^{(k-1)} \cup G_{j^{(k)}}$. Let $\beta^{(k)} = \hat{\beta}_X(\mathcal{G}^{(k)}, y)$.
**End**

Figure 1: Method *Group-OMP*

# 3 Consistency Results

## 3.1 Notation

Let $\mathcal{G}_{\text{good}}$ denote the set of all the groups included in the true model. We refer to the groups in $\mathcal{G}_{\text{good}}$ as good groups. Similarly we call $\mathcal{G}_{\text{bad}}$ the set of all the groups which are not included. We let $g_{\text{good}}$ and $g_{\text{bad}}$ denote the set of "good incides" and "bad indices", i.e. $g_{\text{good}} = \bigcup_{G_i \in \mathcal{G}_{\text{good}}} G_i$ and $g_{\text{bad}} = \bigcup_{G_i \in \mathcal{G}_{\text{bad}}} G_i$. When they are used to restrict index sets for matrix columns or vectors, they are assumed to be in canonical (ascending) order, as we did for $G$. Furthermore, the elements of $\mathcal{G}_{\text{good}}$ are groups of indices, and $|\mathcal{G}_{\text{good}}|$ is the number of groups in $\mathcal{G}_{\text{good}}$, while $g_{\text{good}}$ is defined in terms of individual indices, i.e. $g_{\text{good}}$ is the set of indices corresponding to the groups in $\mathcal{G}_{\text{good}}$. The same holds for $\mathcal{G}_{\text{bad}}$ and $g_{\text{bad}}$. In this notation $\text{supp}(\bar{\beta}) \subset g_{\text{good}}$.

We denote by $\rho_X(\mathcal{G}_{\text{good}})$ the smallest eigenvalue of $X^*_{g_{\text{good}}} X_{g_{\text{good}}}$, i.e.
$$\rho_X(\mathcal{G}_{\text{good}}) = \inf_\beta \left\{ \|X\beta\|_2^2 / \|\beta\|_2^2 : \text{supp}(\beta) \subset g_{\text{good}} \right\}.$$

Here and throughout the paper we let $A^*$ denote the conjugation of the matrix $A$ (which, for a real matrix $A$, coincides with its transpose) and $A^+$ denote the Moore–Penrose pseudoinverse of the matrix $A$ (c.f. [6, 7]). If rows of $A$ are linearly independent $A^+ = A^*(AA^*)^{-1}$ and when columns of $A$ are linearly independent $A^+ = (A^*A)^{-1}A^*$. Generally for $u = \{u_1, \ldots, u_{|g_{\text{good}}|}\}$, $v = \{v_1, \ldots, v_{|g_{\text{bad}}|}\}$ we define
$$\|u\|_{(2,1)}^{\text{good}} = \sum_{G_i \in \mathcal{G}_{\text{good}}} \sqrt{\sum_{j \in G_i} u_j^2}, \text{ and } \|v\|_{(2,1)}^{\text{bad}} = \sum_{G_i \in \mathcal{G}_{\text{bad}}} \sqrt{\sum_{j \in G_i} v_j^2}$$
and then for any matrix $A \in \mathbb{R}^{|g_{\text{good}}| \times |g_{\text{bad}}|}$, let $\|A\|_{(2,1)}^{\text{good/bad}} = \sup_{\|v\|_{(2,1)}^{\text{bad}}=1} \|Av\|_{(2,1)}^{\text{good}}$.

Then we define $\mu_X(\mathcal{G}_{\text{good}}) = \|X^+_{g_{\text{good}}} X_{g_{\text{bad}}}\|_{(2,1)}^{\text{good/bad}}$.

## 3.2 The Noiseless Case

We first focus on the noiseless case (i.e. $\nu \equiv 0$). For all $k$, let $r_k = X\beta^{(k)} - y$. In the noiseless case, we have $r_0 = -y \in Span(\mathcal{G}_{\text{good}})$. So if Group-OMP has not made a mistake up to round $k$, we also have $r_k \in Span(\mathcal{G}_{\text{good}})$. The following theorem and its corollary provide a condition which guarantees that Group-OMP does not make a mistake at the next iteration, given that it has not made any mistakes up to that point. By induction on $k$, it implies that Group-OMP never makes a mistake.

**Theorem 1.** *Reorder the groups in such a way that $\mathcal{G}_{\text{good}} = G_1, \ldots, G_m$ and $\mathcal{G}_{\text{bad}} = G_{m+1}, \ldots, G_J$. Let $r \in Span(X_{g_{\text{good}}})$. Then the following holds*

$$\frac{\|(\|X^*_{G_{m+1}} r\|_2, \|X^*_{G_{m+2}} r\|_2, \ldots, \|X^*_{G_J} r\|_2)\|_\infty}{\|(\|X^*_{G_1} r\|_2, \|X^*_{G_2} r\|_2, \ldots, \|X^*_{G_m} r\|_2)\|_\infty} \leq \mu_X(\mathcal{G}_{\text{good}}). \tag{1}$$

*Proof of Theorem 1.* Reorder the groups in such way that $\mathcal{G}_{\text{good}} = \{G_1, \ldots, G_m\}$ and $\mathcal{G}_{\text{bad}} = \{G_{m+1}, \ldots, G_J\}$. Let $\Phi^* : \mathbb{R}^n \to \mathbb{R}^{d_1} \oplus \mathbb{R}^{d_2} \oplus \ldots \oplus \mathbb{R}^{d_m}$ be defined as
$$\Phi^*(x) = \left((X^*_{G_1} x)^T, (X^*_{G_2} x)^T, \ldots, (X^*_{G_m} x)^T\right)^T$$
and analogously let $\Psi^* : \mathbb{R}^n \to \mathbb{R}^{d_{m+1}} \oplus \mathbb{R}^{d_{m+2}} \oplus \ldots \oplus \mathbb{R}^{d_J}$ be defined as
$$\Psi^*(x) = \left((X^*_{G_{m+1}} x)^T, (X^*_{G_{m+2}} x)^T, \ldots, (X^*_{G_J} x)^T\right)^T.$$
We shall denote $V^\Phi = \mathbb{R}^{d_1} \oplus \mathbb{R}^{d_2} \oplus \ldots \oplus \mathbb{R}^{d_m}$ with a norm $\|.\|_{(2,\infty)}^\Phi$ defined as: $\|(v_1, v_2, \ldots, v_m)\|_{(2,\infty)}^\Phi = \|(\|v_1\|_2, \|v_2\|_2, \ldots, \|v_m\|_2)\|_\infty$ for $v_i \in \mathbb{R}^{d_i}$, $i = 1, \ldots, m$. Analogously $V^\Psi = \mathbb{R}^{d_{m+1}} \oplus \mathbb{R}^{d_{m+2}} \oplus \ldots \oplus \mathbb{R}^{d_J}$ with a norm $\|.\|_{(2,\infty)}^\Psi$ defined as: $\|(v_1, v_2, \ldots, v_{J-m})\|_{(2,\infty)}^\Psi = \|(\|v_1\|_2, \|v_2\|_2, \ldots, \|v_{J-m}\|_2)\|_\infty$ for $v_j \in \mathbb{R}^{d_{m+j}}$, $j = 1, \ldots, J - m$. It is easy to verify that $\|.\|_{(2,\infty)}^\Phi$, $\|.\|_{(2,\infty)}^\Psi$ are norms indeed. Now the condition expressed by Eq. (1) can be rephrased as
$$\frac{\|\Psi^*(r)\|_{(2,\infty)}^\Psi}{\|\Phi^*(r)\|_{(2,\infty)}^\Phi} \leq \mu_X(\mathcal{G}_{\text{good}}) \tag{2}$$

**Lemma 1.** *The map $\Phi^*$ restricted to $Span \bigcup_{i=1}^{m} X_{G_i}$ is a linear isomorphism onto its image.*

*Proof of Lemma 1.* By definition if $\Phi^*(x) = (\mathbf{0})_{\mathbf{V}^\Phi}$ then $x$ must be orthogonal to each of the subspaces spanned by $X_{G_i}, i = 1, \ldots, m$. Thus $\ker \Phi^* \cap Span \bigcup_{i=1}^{m} X_{G_i} = \mathbf{0}$ $\qquad\square$

Let $(\Phi^*)^+$ denote the inverse mapping whose existence was proved in Lemma 1. The choice of symbol is not coincident, the matrix of this mapping is indeed a pseudoinverse of the matrix $(X_{G_1} | X_{G_2} | \ldots | X_{G_m})^T$. We have $\frac{\|\Psi^*(r)\|_{(2,\infty)}^{\Psi}}{\|\Phi^*(r)\|_{(2,\infty)}^{\Phi}} = \frac{\|\Psi^*((\Phi^*)^+ \Phi^*(r))\|_{(2,\infty)}^{\Psi}}{\|\Phi^*(r)\|_{(2,\infty)}^{\Phi}} \leq \|\Psi^* \circ (\Phi^*)^+\|_{(2,\infty)}$, where the last term is the norm of the operator $\Psi^* \circ (\Phi^*)^+ : V^\Phi \to V^\Psi$. We are going to need the following

**Lemma 2.** *A dual space of $V^\Phi$ is $(V^\Phi)^* = \mathbb{R}^{d_1} \oplus \mathbb{R}^{d_2} \oplus \ldots \oplus \mathbb{R}^{d_m}$ with a norm $\|.\|_{(2,1)}^{\Phi}$ defined as:* $\|(v_1, v_2, \ldots, v_m)\|_{(2,1)}^{\Phi} = \|(\|v_1\|_2, \|v_2\|_2, \ldots, \|v_m\|_2)\|_1$.
*A dual space of $V^\Psi$ is $(V^\Psi)^* = \mathbb{R}^{d_{m+1}} \oplus \mathbb{R}^{d_{m+2}} \oplus \ldots \oplus \mathbb{R}^{n_J}$ with a norm $\|.\|_{(2,1)}^{\Psi}$ defined as:*
$\|(v_1, v_2, \ldots, v_{J-m})\|_{(2,1)}^{\Psi} = \|(\|v_1\|_2, \|v_2\|_2, \ldots, \|v_{J-m}\|_2)\|_1$.

*Proof of Lemma 2.* We prove for $V^\Psi$, the proof for $V^\Phi$ is identical.
Let $v^* = (v_1^*, v_2^*, \ldots, v_{J-m}^*) \in \mathbb{R}^{d_{m+1}} \oplus \mathbb{R}^{d_{m+2}} \oplus \ldots \oplus \mathbb{R}^{d_J}$. We have

$$\|v^*\| = \sup_{\substack{v \in V^\Psi \\ \|v\|_{2,\infty}=1}} |v^*(v)| = \sup_{\substack{v_i \in \mathbb{R}^{n_i} \\ \|v\|_{2,\infty}=1}} \sum_{i=m+1}^{J} |\langle v_i^*, v_i \rangle| = \sum_{i=m+1}^{J} \sup_{\substack{v_i \in \mathbb{R}^{n_i} \\ \|v_i\|_2 = 1}} |\langle v_i^*, v_i \rangle| = \sum_{i=m+1}^{J} \|v_i^*\|_2.$$

The last equality follows from $\sup_{\substack{v_i \in \mathbb{R}^{n_i} \\ \|v_i\|_2=1}} |\langle v_i^*, v_i \rangle| = \|v_i^*\|_2$ (as $\ell_2^* = \ell_2$) and Schwartz inequality. $\quad\square$

A fundamental fact from Functional Analysis states that a (Hermitian) conjugation is an isometric isomorphism. Thus

$$\|\Psi^* \circ (\Phi^*)^+\|_{(2,\infty)} = \|(\Phi)^+ \circ \Psi\|_{(2,1)}. \tag{3}$$

We used here $(A^*)^* = A$ and $(A^*)^+ = (A^+)^*$. The right hand side of (3) is equal to $\|X_{g_{good}}^+ X_{g_{bad}}\|_{(2,1)}^{good/bad}$ in matrix notation. Thus the inequality (1) holds. This concludes the proof of Theorem 1. $\qquad\square$

**Corollary 1.** *Under the conditions of Theorem 1, if $\mu_X(\mathcal{G}_{good}) < 1$ then the following holds*

$$\frac{\|(\|X_{G_{m+1}}^* r\|_2, \|X_{G_{m+2}}^* r\|_2, \ldots, \|X_{G_J}^* r\|_2)\|_\infty}{\|(\|X_{G_1}^* r\|_2, \|X_{G_2}^* r\|_2, \ldots, \|X_{G_m}^* r\|_2)\|_\infty} < 1. \tag{4}$$

Intuitively, the condition $\mu_X(\mathcal{G}_{good}) < 1$ guarantees that no bad group "mimics" any good group too well. Note that Theorem 1 and Corollary 1 are the counterpart to Theorem 3.3 in [9] which states the Exact Recovery condition for the standard OMP algorithm, namely that $\|X_{g_{good}}^+ X_{g_{bad}}\|_{(1,1)} < 1$, where $g_{good}$ is not defined in terms of groups, but rather in terms of the variables present in the true model (since the notion of groups does not pertain to OMP in its original form).

## 3.3 The Noisy Case

The following theorem extends the results of Theorem 1 to deal with the non-zero Gaussian noise $\nu$. It shows that under certain conditions the Group-OMP algorithm does not select bad groups. A sketch of the proof is provided at the end of this section.

**Theorem 2.** *Assume that $\mu_X(\mathcal{G}_{good}) < 1$ and $1 \geq \rho_X(\mathcal{G}_{good}) > 0$. For any $\eta \in (0, 1/2)$, with probability at least $1 - 2\eta$, if the stopping criterion of the* Group-OMP *algorithm is such that*

$$\epsilon > \frac{1}{1 - \mu_X(\mathcal{G}_{good})} \sigma \sqrt{2d \ln(2d/\eta)},$$

*then when the algorithm stops all of the following hold:*
(C1)$\mathcal{G}^{(k-1)} \subset \mathcal{G}_{good}$

$$(\text{C2}) \|\beta^{(\mathrm{k}-1)} - \hat{\beta}_{\mathrm{X}}(\mathcal{G}_{\mathrm{good}}, \mathrm{y})\|_2 \le \epsilon \frac{\sqrt{|\mathcal{G}_{\mathrm{good}} \setminus \mathcal{G}^{(\mathrm{k}-1)}|}}{\rho_{\mathrm{X}}(\mathcal{G}_{\mathrm{good}})}$$

$$(\text{C3}) \|\hat{\beta}_{\mathrm{X}}(\mathcal{G}_{\mathrm{good}}, \mathrm{y}) - \bar{\beta}\|_\infty \le \sigma \sqrt{\frac{2 \ln(2|\mathrm{g}_{\mathrm{good}}|/\eta)}{\rho_{\mathrm{X}}(\mathcal{G}_{\mathrm{good}})}}$$

$$(\text{C4}) |\mathcal{G}_{\mathrm{good}} \setminus \mathcal{G}^{(\mathrm{k}-1)}| \le 2 \left| \left\{ \mathrm{G_j} \in \mathcal{G}_{\mathrm{good}} : \|\bar{\beta}_{\mathrm{G_j}}\|_2 < \sqrt{8} \epsilon \rho_{\mathrm{X}}(\mathcal{G}_{\mathrm{good}})^{-1} \right\} \right|.$$

We thus obtain the following theorem which states the main consistency result for Group-OMP.

**Theorem 3.** *Assume that $\mu_X(\mathcal{G}_{\mathrm{good}}) < 1$ and $1 \ge \rho_X(\mathcal{G}_{\mathrm{good}}) > 0$. For any $\eta \in (0, 1/2)$, with probability at least $1 - 2\eta$, if the stopping criterion of the* Group-OMP *algorithm is such that $\epsilon > \frac{1}{1 - \mu_X(\mathcal{G}_{\mathrm{good}})} \sigma \sqrt{2d \ln(2d/\eta)}$ and $\min_{G_j \in \mathcal{G}_{\mathrm{good}}} \|\bar{\beta}_{G_j}\|_2 \ge \sqrt{8}\epsilon \rho_X(\mathcal{G}_{\mathrm{good}})^{-1}$ then when the algorithm stops $\mathcal{G}^{(k-1)} = \mathcal{G}_{\mathrm{good}}$ and $\|\beta^{(k-1)} - \bar{\beta}\|_\infty \le \sigma \sqrt{(2 \ln(2|\mathcal{G}_{\mathrm{good}}|/\eta))/\rho_X(\mathcal{G}_{\mathrm{good}})}$.*

Except for the condition on $\mu_X(\mathcal{G}_{\mathrm{good}})$ (and the definition of $\mu_X(\mathcal{G}_{\mathrm{good}})$ itself), the conditions in Theorem 2 and Theorem 3 are similar to those required for the standard OMP algorithm [13], the main advantage being that for Group-OMP it is the $l_2$ norm of the coefficient *groups* for the true model that need to be lower-bounded, rather than the amplitude of the individual coefficients.[1]

*Proof Sketch of Theorem 2.* To prove the theorem a series of lemmas are needed, whose proofs are omitted due to space constraint, as they can be derived using arguments similar to Zhang [13] for the standard OMP case. The following lemma gives a lower bound on the correlation between the good groups and the residuals from the OLS prediction where the coefficients have been restricted to a set of good groups.

**Lemma 3.** *Let $\mathcal{G} \subset \mathcal{G}_{\mathrm{good}}$, i.e., $\mathcal{G}$ is a set of good groups. Let $\beta = \hat{\beta}_X(\mathcal{G}, y)$, $\beta' = \hat{\beta}_X(\mathcal{G}_{\mathrm{good}}, y)$, $f = X\beta$ and $f' = X\beta'$. Then $\max_{G_j \in \mathcal{G}_{\mathrm{good}}} \|X_{G_j}^*(y - f)\|_2 \ge \frac{\sqrt{\rho_X(\mathcal{G}_{\mathrm{good}})}}{\sqrt{|\mathcal{G}_{\mathrm{good}} \setminus \mathcal{G}|}} \|f - f'\|_2.$*

The following lemma relates the parameter $\hat{\beta}_X(\mathcal{G}_{\mathrm{good}})$, which is estimated by OLS given that the set of good groups has been correctly identified, to the true parameter $\bar{\beta}$.

**Lemma 4.** *For all $\eta \in (0, 1)$, with probability at least $1 - \eta$, we have*
$$\|\hat{\beta}_X(\mathcal{G}_{\mathrm{good}}, y) - \hat{\beta}_X(\mathcal{G}_{\mathrm{good}}, \mathbb{E}y)\|_\infty \le \sigma \sqrt{\frac{2 \ln(2|g_{\mathrm{good}}|/\eta)}{\rho_X(\mathcal{G}_{\mathrm{good}})}}.$$

The following lemma provides an upper bound on the correlation of the bad features to the residuals from the prediction by OLS given that the set of good groups has been correctly identified.

**Lemma 5.** *Let $\beta' = \hat{\beta}_X(\mathcal{G}_{\mathrm{good}}, y)$ and $f' = X\beta'$. We have*
$$P\left( \max_{G_j \notin \mathcal{G}_{\mathrm{good}}} \|X_{G_j}^*(f' - y)\|_2 \le \sigma \sqrt{2d \ln(2d/\eta)} \right) \ge 1 - \eta.$$

We are now ready to prove Theorem 2. We first prove that for each iteration $k$ before the *Group-OMP* algorithm stops, $\mathcal{G}^{(k-1)} \subset \mathcal{G}_{\mathrm{good}}$ by induction on $k$. Now, suppose that the claim holds after $k - 1$ iterations, where $k \ge 1$. So at the beginning of the $k$th iteration, we have $\mathcal{G}^{(k-1)} \subset \mathcal{G}_{\mathrm{good}}$. We have

$$\max_{G_j \notin \mathcal{G}_{\mathrm{good}}} \|X_{G_j}^*(X\beta^{(k-1)} - y)\|_2$$

$$\le \max_{G_j \notin \mathcal{G}_{\mathrm{good}}} \|X_{G_j}^* X(\beta^{(k-1)} - \beta')\|_2 + \max_{G_j \notin \mathcal{G}_{\mathrm{good}}} \|X_{G_j}^*(X\beta' - y)\|_2$$

$$\le \mu_X(\mathcal{G}_{\mathrm{good}}) \max_{G_j \in \mathcal{G}_{\mathrm{good}}} \|X_{G_j}^* X(\beta^{(k-1)} - \beta')\|_2 + \max_{G_j \notin \mathcal{G}_{\mathrm{good}}} \|X_{G_j}^*(X\beta' - y)\|_2 \qquad (5)$$

$$= \mu_X(\mathcal{G}_{\mathrm{good}}) \max_{G_j \in \mathcal{G}_{\mathrm{good}}} \|X_{G_j}^*(X\beta^{(k-1)} - y)\|_2 + \max_{G_j \notin \mathcal{G}_{\mathrm{good}}} \|X_{G_j}^*(X\beta' - y)\|_2 \qquad (6)$$

Here Eq. 5 follows by applying Theorem 1, and Eq. 6 is due to the fact that for all $G_j \in \mathcal{G}_{\text{good}}$ $X_{G_j}^*(X\beta' - y) = \mathbf{0}_{(\mathbf{d_j})}$ holds.

Lemma 5 together with the condition on $\epsilon$ of Theorem 2 implies that with probability at least $1 - \eta$,

$$\max_{G_j \notin \mathcal{G}_{\text{good}}} \|X_{G_j}^*(X\beta' - y)\|_2 \leq \sigma\sqrt{2d\ln(2d/\eta)} < (1 - \mu_X(\mathcal{G}_{\text{good}}))\epsilon. \qquad (7)$$

Lemma 3 together with the definition of $\rho_X(\mathcal{G}_{\text{good}})$ implies

$$\max_{G_j \in \mathcal{G}_{\text{good}}} \|X_{G_j}^*(y - X\beta^{(k-1)})\|_2 \geq \frac{\rho_X(\mathcal{G}_{\text{good}})}{\sqrt{|\mathcal{G}_{\text{good}} \setminus \mathcal{G}^{(k-1)}|}} \|\beta^{(k-1)} - \beta'\|_2 \qquad (8)$$

We then have to deal with the following cases.

**Case 1:** $\|\beta^{(k-1)} - \beta'\|_2 > \epsilon \frac{\sqrt{|\mathcal{G}_{\text{good}} \setminus \mathcal{G}^{(k-1)}|}}{\rho_X(\mathcal{G}_{\text{good}})}$. It follows that

$$\max_{G_j \in \mathcal{G}_{\text{good}}} \|X_{G_j}^*(y - X\beta^{(k-1)})\|_2 > \epsilon > \max_{G_j \notin \mathcal{G}} \|X_{G_j}^*(X\beta' - y)\|_2/(1 - \mu_X(\mathcal{G}_{\text{good}})), \qquad (9)$$

where the last inequality follows from Eq. 7. Then Eq. 6 implies that $\max_{G_j \notin \mathcal{G}_{\text{good}}} \|X_{G_j}^*(X\beta^{(k-1)} - y)\|_2 < \max_{G_j \in \mathcal{G}_{\text{good}}} \|X_{G_j}^*(X\beta^{(k-1)} - y)\|_2$. So a good group is selected, i.e., $G_{i(k)} \in \mathcal{G}_{\text{good}}$ and Eq. 9 implies that the algorithm does not stop.

**Case 2:** $\|\beta^{(k-1)} - \beta'\|_2 \leq \epsilon \frac{\sqrt{|\mathcal{G}_{\text{good}} \setminus \mathcal{G}^{(k-1)}|}}{\rho_X(\mathcal{G}_{\text{good}})}$. We then have three possibilities.
**Case 2.1:** $G_{i(k)} \in \mathcal{G}_{\text{good}}$ and the procedure does not stop.
**Case 2.2:** $G_{i(k)} \in \mathcal{G}_{\text{good}}$ and the procedure stops.
**Case 2.3:** $G_{i(k)} \notin \mathcal{G}_{\text{good}}$ in which case we have $\max_{G_j \in \mathcal{G}_{\text{good}}} \|X_{G_j}^*(X\beta^{(k-1)} - y)\|_2 \leq \max_{G_j \notin \mathcal{G}_{\text{good}}} \|X_{G_j}^*(X\beta^{(k-1)} - y)\|_2 \leq \mu_X(\mathcal{G}_{\text{good}}) \max_{G_j \in \mathcal{G}_{\text{good}}} \|X_{G_j}^*(X\beta^{(k-1)} - y)\|_2 + \max_{G_j \notin \mathcal{G}_{\text{good}}} \|X_{G_j}^*(X\beta' - y)\|_2 \leq \mu_X(\mathcal{G}_{\text{good}}) \max_{G_j \notin \mathcal{G}_{\text{good}}} \|X_{G_j}^*(X\beta^{(k-1)} - y)\|_2 + \max_{G_j \notin \mathcal{G}_{\text{good}}} \|X_{G_j}^*(X\beta' - y)\|_2$, where the second inequality follows from Eq. 6 and the last follows from applying the first inequality once again. We thus obtain that $\max_{G_j \notin \mathcal{G}_{\text{good}}} \|X_{G_j}^*(X\beta^{(k-1)} - y)\|_2 \leq \frac{1}{1 - \mu_X(\mathcal{G}_{\text{good}})} \max_{G_j \notin \mathcal{G}_{\text{good}}} \|X_{G_j}^*(X\beta' - y)\|_2 < \epsilon$, where the last inequality follows by Eq. 7. Hence the algorithm stops.

The above cases imply that if the algorithm does not stop we have $G_{i(k)} \in \mathcal{G}_{\text{good}}$, and hence $\mathcal{G}^{(k)} \subseteq \mathcal{G}_{\text{good}}$ and if the algorithm stops we have $\|\beta^{(k-1)} - \beta'\|_2 \leq \epsilon \frac{\sqrt{|\mathcal{G}_{\text{good}} \setminus \mathcal{G}^{(k-1)}|}}{\rho_X(\mathcal{G}_{\text{good}})}$. Thus by induction, if the *Group-OMP* algorithm stops at iteration $k$, we have that $\mathcal{G}^{(k-1)} \subseteq \mathcal{G}_{\text{good}}$ and $\|\beta^{(k-1)} - \beta'\|_2 \leq \epsilon \frac{\sqrt{|\mathcal{G}_{\text{good}} \setminus \mathcal{G}^{(k-1)}|}}{\rho_X(\mathcal{G}_{\text{good}})}$. So (C1) and (C2) are satisfied. Lemma 4 implies that (C3) holds, and together with the theorem's condition on $\epsilon$ also implies that with probability at least $1 - \eta$, we have $\|\hat{\beta}_X(\mathcal{G}_{\text{good}}, y) - \hat{\beta}_X(\mathcal{G}_{\text{good}}, \mathbb{E}y)\|_\infty \leq \sigma\sqrt{(2\ln(2|\mathcal{G}_{\text{good}}|/\eta))/\rho_X(\mathcal{G}_{\text{good}})} < \epsilon/\sqrt{\rho_X(\mathcal{G}_{\text{good}})}$. This allows us to show that (C4) holds, using similar arguments as in [13], which we omit due to space constraints. This leads to Theorem 2. $\qquad\square$

# 4 Experiments

## 4.1 Simulation Results

We empirically evaluate the performance of the proposed Group-OMP method, against comparison methods OMP, Group Lasso, Lasso and OLS (Ordinary Least Square). Comparison with OMP will test the effect of "grouping" OMP, while Group Lasso is included as a representative existing method of group variable selection. We compare the performance of these methods in terms of the accuracy of variable selection, variable *group* selection and prediction. As measure of variable (group) selection accuracy we use the $F_1$ measure, which is defined as $F_1 = \frac{2PR}{P+R}$, where $P$ denotes the precision and $R$ denotes the recall. For computing variable group $F_1$ for a variable selection method,

we consider a group to be selected if *any* of the variables in the group is selected.[2] As measure of prediction accuracy, we use the *model error*, defined as Model error = $(\hat{\beta} - \bar{\beta})^* E(X^*X)(\hat{\beta} - \bar{\beta})$, where $\bar{\beta}$ are the true model coefficients and $\hat{\beta}$ the estimated coefficients. Recall that Lasso solves $\arg\min_\beta \left( \|Y - X\beta\|^2 + \lambda\|\beta\|_1 \right)$. So the tuning parameter for Lasso and Group Lasso is the penalty parameter $\lambda$. For Group-OMP and OMP rather than parameterizing the models according to precision $\epsilon$, we do so using the iteration number (i.e. a stopping point). We consider two estimates: the "oracle estimate" and the "holdout validated estimate". For the oracle estimate, the tuning parameter is chosen so as to minimize the model error. Note that such estimate can only be computed in simulations and not in practical situations, but it is useful for evaluating the relative performance of comparison methods, independently of the appropriateness of the complexity parameter. The holdout-validated estimate is a practical version of the oracle estimate, obtained by selecting the tuning parameter by minimizing the average squared error on a validation set. We now describe the experimental setup.

**Experiment 1:** We use an additive model with categorical variables taken from [12](model I). Consider variables $Z_1, \ldots, Z_{15}$, where $Z_i \sim \mathcal{N}(0,1)(i = 1, \ldots, 15)$ and $\text{cov}(Z_i, Z_j) = 0.5^{|i-j|}$. Let $W_1, \ldots, W_{15}$ be such that $W_i = 0$ if $Z_i < \Phi^{-1}(1/3)$, $W_i = 1$ if $Z_i > \Phi^{-1}(2/3)$ and $W_i = 2$ if $\Phi^{-1}(1/3) \le Z_i \le \Phi^{-1}(2/3)$, where $\Phi^{-1}$ is the quantile function for the normal distribution. The responses in the data are generated using the true model:
$Y = 1.8I(W_1 = 1) - 1.2I(W_1 = 0) + I(W_3 = 1) + 0.5I(W_3 = 0) + I(W_5 = 1) + I(W_5 = 0) + \nu$, where $I$ denote the indicator function and $\nu \sim \mathcal{N}(0, \sigma = 1.476)$. Then let $(X_{2(i-1)+1}, X_{2i}) = (I(W_i = 1), I(W_i = 0))$, which are the variables that the estimation methods use as the explanatory variables, with the following variable groups: $G_i = \{2i - 1, 2i\}(i = 1, \ldots, 15)$. We ran 100 runs, each with 50 observations for training and 25 for validation.

**Experiment 2:** We use an additive model with continuous variables taken from [12](model III), where the groups correspond to the expansion of each variable into a third-order polynomial. . Consider variables $Z_1, \ldots, Z_{17}$, with $Z_i$ i.i.d. $\sim \mathcal{N}(0,1)$ $(i = 1, \ldots, 17)$. Let $W_1, \ldots, W_{16}$ be defined as $W_i = (Z_i + Z_{17})/\sqrt{2}$. The true model is $Y = W_3^3 + W_3^2 + W_3 + \frac{1}{3}W_6^3 - W_6^2 + \frac{2}{3}W_6 + \nu$, where $\nu \sim \mathcal{N}(0, \sigma = 2)$. Then let the explanatory variables be $(X_{3(i-1)+1}, X_{3(i-1)+2}, X_{3i}) = \left(W_i^3, W_i^2, W_i\right)$ with the variable groups $G_i = \{3(i-1) + 1, 3(i-1) + 2, 3i\}(i = 1, \ldots, 16)$. We ran 100 runs, each with 100 observations for training and 50 for validation.

**Experiment 3:** We use an additive model with continuous variables similar to that of [16]. Consider three independent hidden variables $Z_1, \ldots, Z_3$ such that $Z_i \sim \mathcal{N}(0, \sigma = 1)$. Consider 40 predictors defined as: $X_i = Z_{\lfloor (i-1)/3 \rfloor + 1} + \nu_i$ for $i = 1, \ldots, 15$ and $X_i \sim \mathcal{N}(0,1)$ for $i = 16, \ldots, 40$, where $\nu_i$ i.i.d. $\sim \mathcal{N}(0, \sigma = 0.1^{1/2})$. The true model is
$\qquad Y = 3\sum_{i=1}^{5} X_i + 4\sum_{i=6}^{10} X_i + 2\sum_{i=11}^{15} X_i + \nu$, where $\nu \sim \mathcal{N}(0, \sigma = 15)$
and the groups are $G_k = \{5(k-1) + 1, \ldots, 5k\}$, for $k = (1, \ldots, 3)$, and $G_k = k + 12$, for $k > 3$. We ran 100 runs, each with 500 observations for training and 50 for validation.

**Experiment 4:** We use an additive model with continuous variables taken from [15]. Consider five hidden variables $Z_1, \ldots, Z_5$ such that $Z_i$ i.i.d. $\sim \mathcal{N}(0, \sigma = 1)$. Consider 10 measurements of each of these hidden variables such that $X_i = (0.05)Z_{\lfloor (i-1)/10 \rfloor + 1} + (1 - 0.05^2)^{1/2}\nu_i$, i=1,...,50, where $\nu_i \sim \mathcal{N}(0,1)$ and $\text{cov}(\nu_i, \nu_j) = 0.5^{|i-j|}$. The true model is $Y = X\bar{\beta} + \nu$, where $\nu \sim \mathcal{N}(0, \sigma = 19.22)$, and

$$\bar{\beta}_i = \begin{cases} 7 & \text{for } i = 1, \ldots, 10 \\ 2 & \text{for } i = 11, \ldots, 20 \\ 1 & \text{for } i = 21, \ldots, 30 \\ 0 & \text{for } i = 31, \ldots, 50 \end{cases}$$

The groups are $G_k = \{10(k-1) + 1, \ldots, 10k\}$, for $k = (1, \ldots, 5)$. We ran 100 runs, each with 300 observations for training and 50 for validation.

The results of the four experiments are presented in Table 1. We note that $F_1$ (Var) and $F_1$ (Group) are identical for the grouped methods for Experiments 1, 2 and 4, since in these the groups have equal size. Overall, Group-OMP performs consistently better than all the comparison methods, with respect to all measures considered . In particular, Group-OMP does better than OMP not only for

| $F_1$ (Var) | Exp 1 | Exp 2 | Exp 3 | Exp 4 |
|---|---|---|---|---|
| OLS | $0.333 \pm 0$ | $0.222 \pm 0$ | $0.545 \pm 0$ | $0.750 \pm 0$ |
| Lasso (Oracle) | $0.483 \pm 0.010$ | $0.541 \pm 0.010$ | $0.771 \pm 0.007$ | $0.817 \pm 0.004$ |
| Lasso (Holdout) | $0.389 \pm 0.012$ | $0.528 \pm 0.015$ | $0.758 \pm 0.015$ | $0.810 \pm 0.005$ |
| OMP (Oracle) | $0.531 \pm 0.019$ | $0.787 \pm 0.009$ | $0.532 \pm 0.004$ | $0.781 \pm 0.005$ |
| OMP (Holdout) | $0.422 \pm 0.014$ | $0.728 \pm 0.013$ | $0.477 \pm 0.006$ | $0.741 \pm 0.006$ |
| Group Lasso (Oracle) | $0.545 \pm 0.010$ | $0.449 \pm 0.011$ | $0.693 \pm 0.005$ | $0.755 \pm 0.002$ |
| Group Lasso (Holdout) | $\mathbf{0.624 \pm 0.017}$ | $0.459 \pm 0.016$ | $0.706 \pm 0.013$ | $0.794 \pm 0.008$ |
| Group-OMP (Oracle) | $\mathbf{0.730 \pm 0.017}$ | $\mathbf{0.998 \pm 0.002}$ | $\mathbf{0.999 \pm 0.001}$ | $\mathbf{0.998 \pm 0.002}$ |
| Group-OMP (Holdout) | $0.615 \pm 0.020$ | $\mathbf{0.921 \pm 0.012}$ | $\mathbf{0.918 \pm 0.011}$ | $\mathbf{0.890 \pm 0.011}$ |
| $F_1$ (Group) | Exp 1 | Exp 2 | Exp 3 | Exp 4 |
| OLS | $0.333 \pm 0$ | $0.222 \pm 0$ | $0.194 \pm 0$ | $0.750 \pm 0$ |
| Lasso (Oracle) | $0.458 \pm 0.012$ | $0.346 \pm 0.008$ | $0.494 \pm 0.011$ | $0.751 \pm 0.001$ |
| Lasso (Holdout) | $0.511 \pm 0.010$ | $0.340 \pm 0.014$ | $0.547 \pm 0.029$ | $0.776 \pm 0.006$ |
| OMP (Oracle) | $0.687 \pm 0.018$ | $0.808 \pm 0.020$ | $0.224 \pm 0.004$ | $0.842 \pm 0.010$ |
| OMP (Holdout) | $0.621 \pm 0.020$ | $0.721 \pm 0.025$ | $0.421 \pm 0.026$ | $0.827 \pm 0.010$ |
| Group Lasso (Oracle) | $0.545 \pm 0.010$ | $0.449 \pm 0.011$ | $0.317 \pm 0.006$ | $0.755 \pm 0.002$ |
| Group Lasso (Holdout) | $\mathbf{0.624 \pm 0.017}$ | $0.459 \pm 0.016$ | $0.364 \pm 0.018$ | $0.794 \pm 0.008$ |
| Group-OMP (Oracle) | $\mathbf{0.730 \pm 0.017}$ | $\mathbf{0.998 \pm 0.002}$ | $\mathbf{0.998 \pm 0.001}$ | $\mathbf{0.998 \pm 0.002}$ |
| Group-OMP (Holdout) | $0.615 \pm 0.020$ | $\mathbf{0.921 \pm 0.012}$ | $\mathbf{0.782 \pm 0.025}$ | $\mathbf{0.890 \pm 0.011}$ |
| ME | Exp 1 | Exp 2 | Exp 3 | Exp 4 |
| OLS | $3.184 \pm 0.129$ | $7.063 \pm 0.251$ | $19.592 \pm 0.451$ | $46.845 \pm 0.985$ |
| Lasso (Oracle) | $1.203 \pm 0.078$ | $1.099 \pm 0.067$ | $9.228 \pm 0.285$ | $30.343 \pm 0.796$ |
| Lasso (Holdout) | $2.536 \pm 0.097$ | $1.309 \pm 0.080$ | $12.987 \pm 0.670$ | $38.089 \pm 1.353$ |
| OMP (Oracle) | $0.711 \pm 0.020$ | $1.052 \pm 0.061$ | $19.006 \pm 0.443$ | $38.497 \pm 0.926$ |
| OMP (Holdout) | $\mathbf{0.945 \pm 0.031}$ | $1.394 \pm 0.102$ | $28.246 \pm 1.942$ | $48.564 \pm 1.957$ |
| Group Lasso (Oracle) | $\mathbf{0.457 \pm 0.021}$ | $0.867 \pm 0.052$ | $11.538 \pm 0.370$ | $31.053 \pm 0.831$ |
| Group Lasso (Holdout) | $1.279 \pm 0.017$ | $1.047 \pm 0.075$ | $14.979 \pm 0.538$ | $37.359 \pm 1.260$ |
| Group-OMP (Oracle) | $0.601 \pm 0.0273$ | $\mathbf{0.379 \pm 0.035}$ | $\mathbf{6.727 \pm 0.252}$ | $\mathbf{27.765 \pm 0.703}$ |
| Group-OMP (Holdout) | $0.965 \pm 0.050$ | $\mathbf{0.605 \pm 0.089}$ | $\mathbf{12.553 \pm 1.469}$ | $\mathbf{35.989 \pm 1.127}$ |

Table 1: Average $F_1$ score at the variable level and group level, and model error for the models output by Ordinary Least Squares, Lasso, OMP, Group Lasso, and Group-OMP.

| Boston Housing | OLS | Lasso | OMP | Group Lasso | Group-OMP |
|---|---|---|---|---|---|
| Prediction Error | $29.30 \pm 3.25$ | $17.82 \pm 0.48$ | $19.10 \pm 0.78$ | $18.45 \pm 0.59$ | $\mathbf{17.60 \pm 0.51}$ |
| Number of Original Variables | $13 \pm 0$ | $12.82 \pm 0.05$ | $11.51 \pm 0.20$ | $12.50 \pm 0.13$ | $9.09 \pm 0.31$ |

Table 2: Average test set prediction error, average number of original variables, for the models output by OLS, Lasso, OMP, Group Lasso, and Group-OMP on the "Boston Housing" dataset.

variable group selection, but also for variable selection and predictive accuracy. Against Group-Lasso, Group-OMP does better in all four experiments with respect to variable (group) selection when using Oracle, while it does worse in one case when using holdout validation. Group-OMP also does better than Group-Lasso with respect to the model error in three out of the four experiments.

## 4.2 Experiment on a real dataset

We use the "Boston Housing" dataset (UCI Machine Learning Repository). The continuous variables appear to have non-linear effects on the target value, so for each such variable, say $X_i$, we consider its third-order polynomial expansion, i.e., $X_i$, $X_i^2$ and $X_i^3$, and consider them as a variable group. We ran 100 runs, where for each run we select at random half of the instances as training examples, one quarter as validation set, and the remaining quarter as test examples. The penalty parameter was chosen with holdout validation for all methods. The average test set prediction error, the average number of selected original variables (i.e. groups) are reported in Table 2. These results confirm that Group-OMP has the highest prediction accuracy among the comparison methods, and also leads to the sparsest model.

## 5 Concluding Remarks

In addition to its merits in terms of consistency and accuracy, Group-OMP is particulary attractive due to its computational efficiency (the entire path is computed in $J$ rounds, where $J$ is the number of groups). Interesting directions for future research include comparing the conditions for the consistency of Group-OMP to those for Group Lasso and the bounds on their respective accuracy in estimating the regression coefficients, evaluating modified versions of Group-OMP where the group selection step $(*)$ in Figure 1 includes a penalty to account for the group size, and considering a forward/backward extension that allows correcting for mistakes (similarly to [14]).

## Footnotes

[1]The sample size $n$ is explicitly part of the conditions in [13] while it is implicit here due to the different ways of normalizing the matrix $X$. One recovers the same dependency on $n$ by considering $X' = \sqrt{n}X$, $\beta'^{(k)} = \beta^{(k)}/\sqrt{n}$, $\bar{\beta}' = \bar{\beta}/\sqrt{n}$, defining (as in [13]) $\rho'_{X'}(\mathcal{G}_{\mathrm{good}}) = \inf_\beta \left\{ \frac{1}{n} \|X'\beta\|_2^2/\|\beta\|_2^2 : \mathrm{supp}(\beta) \subset \mathrm{g}_{\mathrm{good}} \right\}$, and noting that $\rho'_{X'}(\mathcal{G}_{\mathrm{good}}) = \rho_X(\mathcal{G}_{\mathrm{good}})$ and $\hat{\beta}_{X'}(\mathcal{G}_{\mathrm{good}}, y) = \hat{\beta}_X(\mathcal{G}_{\mathrm{good}}, y)/\sqrt{n}$. If $X$ had i.i.d. entries, with mean 0, variance $1/n$ and finite 4th moment, $\rho_X(\mathcal{G}_{\mathrm{good}})$ converges a.s. to $(1 - \sqrt{g})^2$ as $n \to \infty$ and $|\mathrm{g}_{\mathrm{good}}|/n \to g \le 1$ [2]. Hence the rates in C2-C4 are unaffected by $\rho_X(\mathcal{G}_{\mathrm{good}})$.

[2]Other ways of translating variable selection to variable group selection are possible, but the $F_1$ measure is relatively robust with respect to this choice.

# References

[1] BACH, F.R., *Consistency of the Group Lasso and Multiple Kernel Learning*, J. Mach. Learn. Res., **9**, 1179-1225, 2008.

[2] BAI D., YIN Y.Q., *Limit of the smallest eigenvalue of a large dimensional sample covariance matrix*, Ann. Probab. 21, 1275-1294, 1993.

[3] CHEN J., HUO X., *Sparse representations for multiple measurement vectors (MMV) in an overcomplete dictionary*, in Proc. of the 2005 IEEE Int. Conf. on Acoustics, Speech, and Signal Proc., 2005.

[4] HUANG J., ZHANG T., METAXAS D., *Learning with Structured Sparsity,* in ICML'09, 2009.

[5] MALLAT S., ZHANG Z., *Matching pursuits with time-frequency dictionaries*, IEEE Transactions on Signal Processing, **41**, 3397-3415, 1993.

[6] MOORE, E.H, *On the reciprocal of the general algebraic matrix*, Bulletin of the American Mathematical Society **26**, 394-395, 1920.

[7] PENROSE, R., *A generalized inverse for matrices*, Proceedings of the Cambridge Philosophical Society **51**, 406-413, 1955.

[8] TIBSHIRANI, R., *Regression shrinkage and selection via the lasso*, J. Royal. Statist. Soc B., **58**(1), 267-288, 1996.

[9] TROPP J.A., *Greed is good: Algorithmic results for sparse approximation*, IEEE Trans. Info. Theory, **50**(10), 2231-2242, 2004.

[10] TROPP J.A., GILBERT A.C. , STRAUSS M.J., *Algorithms for simultaneous sparse approximation, Part I: greedy pursuit*, Signal Proc. **86** (3), 572-588, 2006.

[11] PEOTTA L., VANDERGHEYNST P., *Matching Pursuit with Block Incoherent Dictionaries*, Signal Proc. **55** (9), 2007.

[12] YUAN, M., LIN, Y., *Model selection and estimation in regression with grouped variables*, J. R. Statist. Soc. B, **68**, 4967, 2006.

[13] ZHANG, T., *On the consistency of feature selection using greedy least squares regression*, J. Machine Learning Research, 2008.

[14] ZHANG, T., *Adaptive Forward-Backward Greedy Algorithm for Sparse Learning with Linear Models*, in NIPS08, 2008.

[15] ZHAO, P, ROCHA, G. AND YU, B., *Grouped and hierarchical model selection through composite absolute penalties,* Manuscript, 2006.

[16] ZOU, H., HASTIE T., *Regularization and variable selection via the Elastic Net.*, J. R. Statist. Soc. B, **67**(2) 301-320, 2005.

